# A Revolution: Belief Propagation in Graphs With Cycles

**Brendan J. Frey**[*]
http://www.cs.utoronto.ca/~frey
Department of Computer Science
University of Toronto

**David J. C. MacKay**
http://wol.ra.phy.cam.ac.uk/mackay
Department of Physics, Cavendish Laboratory
Cambridge University

## Abstract

Until recently, artificial intelligence researchers have frowned upon the application of probability propagation in Bayesian belief networks that have cycles. The probability propagation algorithm is only exact in networks that are cycle-free. However, it has recently been discovered that the two *best* error-correcting decoding algorithms are actually performing probability propagation in belief networks with cycles.

## 1 Communicating over a noisy channel

Our increasingly wired world demands efficient methods for communicating bits of information over physical channels that introduce errors. Examples of real-world channels include twisted-pair telephone wires, shielded cable-TV wire, fiber-optic cable, deep-space radio, terrestrial radio, and indoor radio. Engineers attempt to correct the errors introduced by the noise in these channels through the use of *channel coding* which adds protection to the information source, so that some channel errors can be corrected. A popular model of a physical channel is shown in Fig. 1. A vector of $K$ information bits $\mathbf{u} = (u_1, \ldots, u_K)$, $u_k \in \{0, 1\}$ is encoded, and a vector of $N$ codeword bits $\mathbf{x} = (x_1, \ldots, x_N)$ is transmitted into the channel. Independent Gaussian noise with variance $\sigma^2$ is then added to each codeword bit,

---

[*]Brendan Frey is currently a Beckman Fellow at the Beckman Institute for Advanced Science and Technology, University of Illinois at Urbana-Champaign.

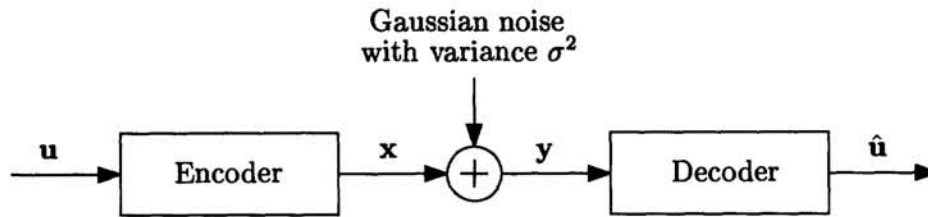

Figure 1: A communication system with a channel that adds Gaussian noise to the transmitted discrete-time sequence.

producing the real-valued channel output vector $\mathbf{y} = (y_1, \ldots, y_N)$. The decoder must then use this received vector to make a guess $\hat{\mathbf{u}}$ at the original information vector. The probability $P_b(e)$ of bit error is minimized by choosing the $u_k$ that maximizes $P(u_k|\mathbf{y})$ for $k = 1, \ldots, K$. The *rate* $K/N$ of a code is the number of information bits communicated per codeword bit. We will consider rate $\sim 1/2$ systems in this paper, where $N = 2K$.

The simplest rate 1/2 encoder duplicates each information bit: $x_{2k-1} = x_{2k} = u_k$, $k = 1, \ldots, K$. The optimal decoder for this *repetition code* simply averages together pairs of noisy channel outputs and then applies a threshold:

$$\hat{u}_k = 1 \quad \text{if } (y_{2k-1} + y_{2k})/2 > 0.5, \quad 0 \text{ otherwise.} \tag{1}$$

Clearly, this procedure has the effect of reducing the noise variance by a factor of $1/2$. The resulting probability $P_b(e)$ that an information bit will be erroneously decoded is given by the area under the tail of the noise Gaussian:

$$P_b(e) = \Phi\left(\frac{-0.5}{\sigma^2/2}\right), \tag{2}$$

where $\Phi()$ is the cumulative standard normal distribution. A plot of $P_b(e)$ versus $\sigma$ for this repetition code is shown in Fig. 2, along with a thumbnail picture that shows the distribution of noisy received signals at the noise level where the repetition code gives $P_b(e) = 10^{-5}$.

More sophisticated channel encoders and decoders can be used to increase the tolerable noise level without increasing the probability of a bit error. This approach can in principle improve performance up to a bound determined by Shannon (1948). For a given probability of bit error $P_b(e)$, this limit gives the maximum noise level that can be tolerated, no matter what channel code is used. Shannon's proof was non-constructive, meaning that he showed that there exist channel codes that achieve his limit, but did not present practical encoders and decoders. The curve for Shannon's limit is also shown in Fig. 2.

The two curves described above define the region of interest for practical channel coding systems. For a given $P_b(e)$, if a system requires a lower noise level than the repetition code, then it is not very interesting. At the other extreme, it is impossible for a system to tolerate a higher noise level than Shannon's limit.

## 2    Decoding Hamming codes by probability propagation

One way to *detect* errors in a string of bits is to add a parity-check bit that is chosen so that the sum modulo 2 of all the bits is 0. If the channel flips one bit, the receiver will find that the sum modulo 2 is 1, and can detect than an error occurred. In a simple Hamming code, the codeword $\mathbf{x}$ consists of the original vector

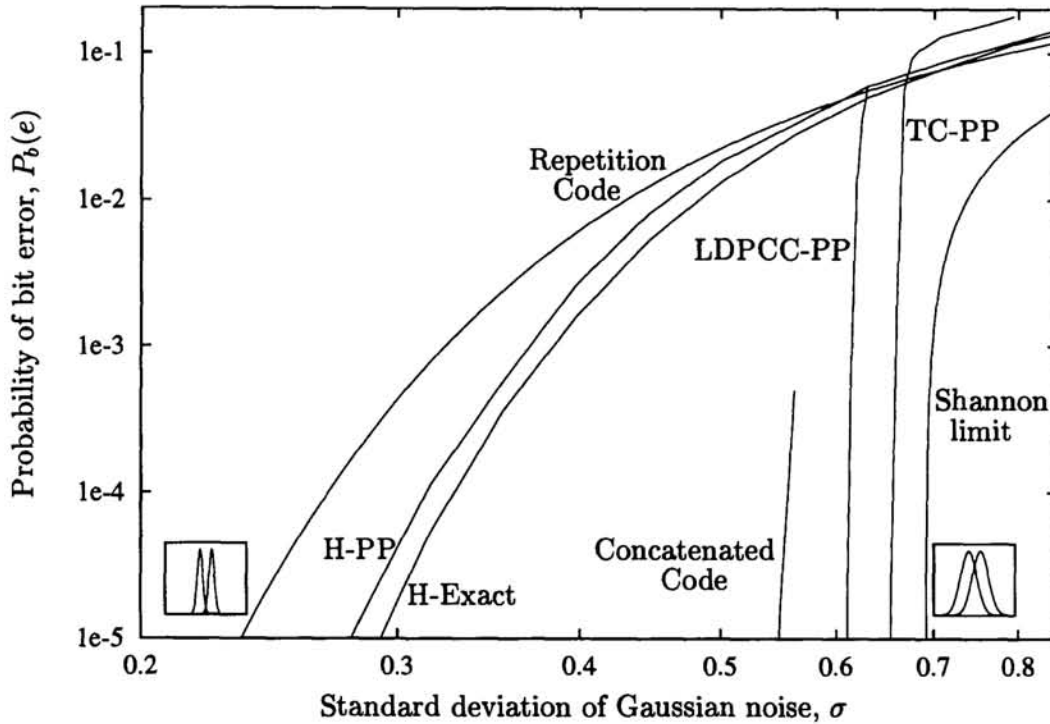

Figure 2: Probability of bit error $P_b(e)$ versus noise level $\sigma$ for several codes with rates near 1/2, using 0/1 signalling. It is impossible to obtain a $P_b(e)$ below Shannon's limit (shown on the far right for rate 1/2). "H-PP" = Hamming code (rate 4/7) decoded by probability propagation (5 iterations); "H-Exact" = Hamming code decoded exactly; "LDPCC-PP" = low-density parity-check coded decoded by probability propagation; "TC-PP" = turbocode decoded by probability propagation. The thumbnail pictures show the distribution of noisy received signals at the noise levels where the repetition code and the Shannon limit give $P_b(e) = 10^{-5}$.

**u** in addition to several parity-check bits, each of which depends on a different *subset* of the information bits. In this way, the Hamming code can not only detect errors but also *correct* them.

The code can be cast in the form of the conditional probabilities that specify a Bayesian network. The Bayesian network for a $K = 4$, $N = 7$ rate 4/7 Hamming code is shown in Fig. 3a. Assuming the information bits are uniformly random, we have $P(u_k) = 0.5$, $u_k \in \{0, 1\}$, $k = 1, 2, 3, 4$. Codeword bits 1 to 4 are direct copies of the information bits: $P(x_k|u_k) = \delta(x_k, u_k)$, $k = 1, 2, 3, 4$, where $\delta(a, b) = 1$ if $a = b$ and 0 otherwise. Codeword bits 5 to 7 are parity-check bits: $P(x_5|u_1, u_2, u_3) = \delta(x_5, u_1 \oplus u_2 \oplus u_3)$, $P(x_6|u_1, u_2, u_4) = \delta(x_6, u_1 \oplus u_2 \oplus u_4)$, $P(x_7|u_2, u_3, u_4) = \delta(x_7, u_2 \oplus u_3 \oplus u_4)$, where $\oplus$ indicates addition modulo 2 (XOR). Finally, the conditional channel probability densities are

$$p(y_n|x_n) = \frac{1}{\sqrt{2\pi\sigma^2}} e^{-(y_n - x_n)^2/2\sigma^2}, \tag{3}$$

for $n = 1, \ldots, 7$.

The probabilities $P(u_k|\mathbf{y})$ can be computed exactly in this belief network, using Lauritzen and Spiegelhalter's algorithm (1988) or just brute force computation. However, for the more powerful codes discussed below, exact computations are intractable. Instead, one way the decoder can approximate the probabilities $P(u_k|\mathbf{y})$ is by applying the probability propagation algorithm (Pearl 1988) to the Bayesian network. Probability propagation is only approximate in this case because the

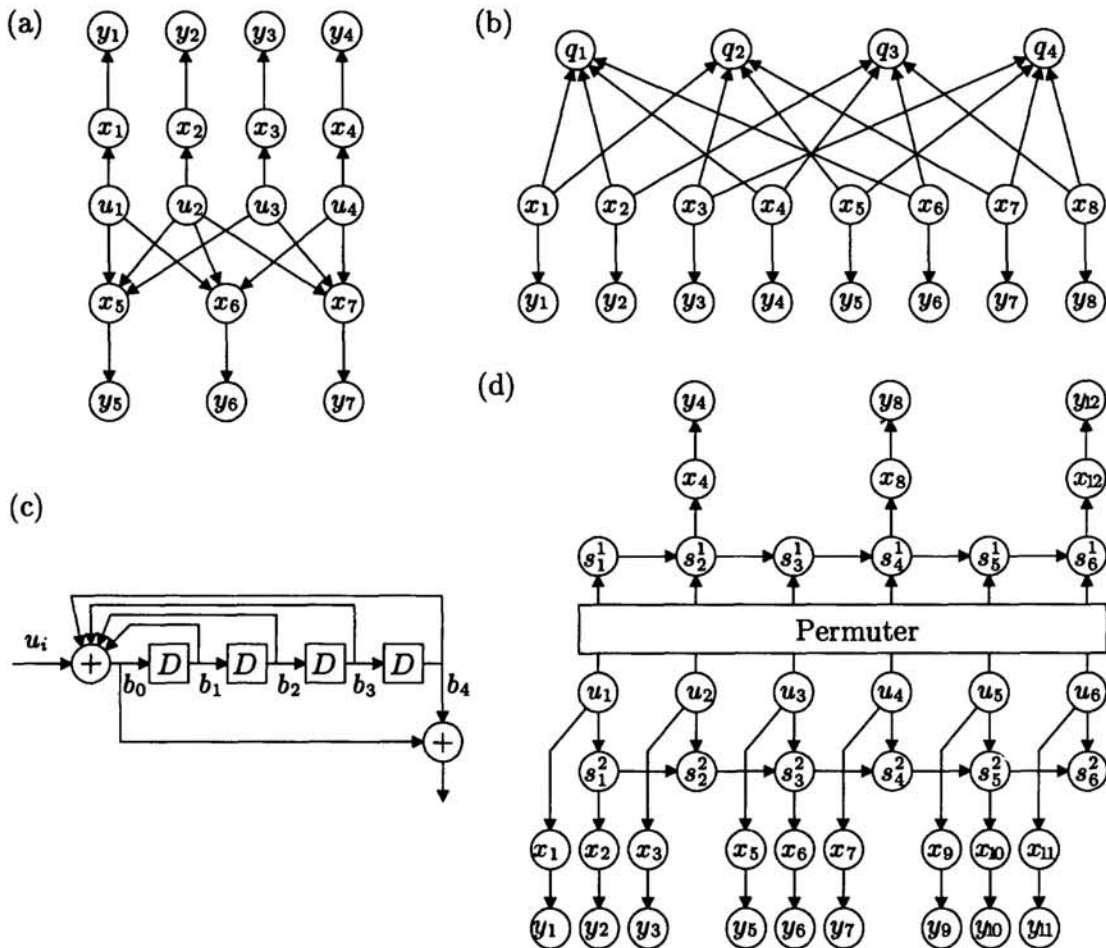

Figure 3: (a) The Bayesian network for a $K = 4$, $N = 7$ Hamming code. (b) The Bayesian network for a $K = 4$, $N = 8$ low-density parity-check code. (c) A block diagram for the turbocode linear feedback shift register. (d) The Bayesian network for a $K = 6$, $N = 12$ turbocode.

network contains cycles (*ignoring* edge directions), *e.g.*, $u_1$-$x_5$-$u_2$-$x_6$-$u_1$. Once a channel output vector **y** is observed, propagation begins by sending a message from $y_n$ to $x_n$ for $n = 1, \ldots, 7$. Then, a message is sent from $x_k$ to $u_k$ for $k = 1, 2, 3, 4$. An *iteration* now begins by sending messages from the information variables $u_1$, $u_2$, $u_3$, $u_4$ to the parity-check variables $x_5$, $x_6$, $x_7$ in parallel. The iteration finishes by sending messages from the parity-check variables back to the information variables in parallel. Each time an iteration is completed, new estimates of $P(u_k|\mathbf{y})$ for $k = 1, 2, 3, 4$ are obtained.

The $P_b(e) - \sigma$ curve for optimal decoding and the curve for the probability propagation decoder (5 iterations) are shown in Fig. 2. Quite surprisingly, the performance of the iterative decoder is quite close to that of the optimal decoder. Our expectation was that short cycles would confound the probability propagation decoder. However, it seems that good performance can be obtained even when there are short cycles in the code network.

For this simple Hamming code, the complexities of the probability propagation decoder and the exact decoder are comparable. However, the similarity in performance between these two decoders prompts the question: "Can probability propagation decoders give performances comparable to exact decoding in cases where exact decoding is computationally intractable?"

# 3 A leap towards the limit: Low-density parity-check codes

Recently, there has been an explosion of interest in the channel coding community in two new coding systems that have brought us a leap closer to Shannon's limit. Both of these codes can be described by Bayesian networks with cycles, and it turns out that the corresponding iterative decoders are performing probability propagation in these networks.

Fig. 3b shows the Bayesian network for a simple *low-density parity-check code* (Gallager 1963). In this network, the information bits are not represented explicitly. Instead, the network defines a set of allowed configurations for the codewords. Each parity-check vertex $q_i$ requires that the codeword bits $\{x_n\}_{n \in Q_i}$ to which $q_i$ is connected have even parity:

$$P(q_i | \{x_n\}_{n \in Q_i}) = \delta(q_i, \bigoplus_{n \in Q_i} x_n), \tag{4}$$

where **q** is clamped to **0** to ensure even parity. Here, $Q_i$ is the set of indices of the codeword bits to which parity-check vertex $q_i$ is connected. The conditional probability densities for the channel outputs are the same as in Eq. 3.

One way to view the above code is as $N$ binary codeword variables along with a set of linear (modulo 2) equations. If in the end we want there to be $K$ degrees of freedom, then the number of linearly independent parity-check equations should be $N - K$. In the above example, there are $N = 8$ codeword bits and 4 parity-checks, leaving $K = 8 - 4 = 4$ degrees of freedom. It is these degrees of freedom that we use to represent the information vector **u**. Because the code is linear, a $K$-dimensional vector **u** can be mapped to a valid **x** simply by multiplying by an $N \times K$ matrix (using modulo 2 addition). This is how an encoder can produce a low-density parity-check codeword for an input vector.

Once a channel output vector **y** is observed, the iterative probability propagation decoder begins by sending messages from **y** to **x**. An iteration now begins by sending messages from the codeword variables **x** to the parity-check constraint variables **q**. The iteration finishes by sending messages from the parity-check constraint variables back to the codeword variables. Each time an iteration is completed, new estimates of $P(x_n | \mathbf{y})$ for $n = 1, \dots, N$ are obtained. After a valid (but not necessarily correct) codeword has been found, or a prespecified limit on the number of iterations has been reached, decoding stops. The estimate of the codeword is then mapped back to an estimate $\hat{\mathbf{u}}$ of the information vector.

Fig. 2 shows the performance of a $K = 32,621$, $N = 65,389$ low-density parity-check code when decoded as described above. (See MacKay and Neal (1996) for details.) It is impressively close to Shannon's limit — significantly closer than the "Concatenated Code" (described in Lin and Costello (1983)) which was considered the best practical code until recently.

# 4 Another leap: Turbocodes

The codeword for a turbocode (Berrou *et al.* 1996) consists of the original information vector, plus two sets of bits used to protect the information. Each of these two sets is produced by feeding the information bits into a linear feedback shift register (LFSR), which is a type of finite state machine. The two sets differ in that one set is produced by a *permuted* set of information bits; *i.e.*, the order of the bits is scrambled in a fixed way before the bits are fed into the LFSR. Fig. 3c shows a block diagram (*not* a Bayesian network) for the LFSR that was used in our experiments.

Each box represents a delay (memory) element, and each circle performs addition modulo 2. When the $k$th information bit arrives, the machine has a state $s_k$ which can be written as a binary string of state bits $b_4 b_3 b_2 b_1 b_0$ as shown in the figure. $b_0$ of the state $s_k$ is determined by the current input $u_k$ and the previous state $s_{k-1}$. Bits $b_1$ to $b_4$ are just shifted versions of the bits in the previous state.

Fig. 3d shows the Bayesian network for a simple turbocode. Notice that each state variable in the two constituent chains depends on the previous state and an information bit. In each chain, every second LFSR output is not transmitted. In this way, the overall rate of the code is $1/2$, since there are $K = 6$ information bits and $N = 6 + 3 + 3 = 12$ codeword bits. The conditional probabilities for the states of the nonpermuted chain are

$$P(s_k^1 | s_{k-1}^1, u_k) = 1 \quad \text{if state } s_k^1 \text{ follows } s_{k-1}^1 \text{ for input } u_k, \quad 0 \text{ otherwise.} \quad (5)$$

The conditional probabilities for the states in the other chain are similar, except that the inputs are permuted. The probabilities for the information bits are uniform, and the conditional probability densities for the channel outputs are the same as in Eq. 3.

Decoding proceeds with messages being passed from the channel output variables to the constituent chains and the information bits. Next, messages are passed from the information variables to the first constituent chain, $s^1$. Messages are passed forward and then backward along this chain, in the manner of the forward-backward algorithm (Smyth *et al.* 1997). After messages are passed from the first chain to the second chain $s^2$, the second chain is processed using the forward-backward algorithm. To complete the iteration, messages are passed from $s^2$ to the information bits.

Fig. 2 shows the performance of a $K = 65,536$, $N = 131,072$ turbocode when decoded as described above, using a fixed number (18) of iterations. (See Frey (1998) for details.) Its performance is significantly closer to Shannon's limit than the performances of both the low-density parity-check code and the textbook standard "Concatenated Code".

## 5   Open questions

We are certainly not claiming that the NP-hard problem (Cooper 1990) of probabilistic inference in general Bayesian networks can be solved in polynomial time by probability propagation. However, the results presented in this paper do show that there are practical problems which can be solved using *approximate* inference in graphs with cycles. Iterative decoding algorithms are using probability propagation in graphs with cycles, and it is still not well understood why these decoders work so well. Compared to other approximate inference techniques such as variational methods, probability propagation in graphs with cycles is unprincipled. How well do more principled decoders work? In (MacKay and Neal 1995), a variational decoder that maximized a lower bound on $\prod_{k=1}^{K} P(u_k | \mathbf{y})$ was presented for low-density parity-check codes. However, it was found that the performance of the variational decoder was *not* as good as the performance of the probability propagation decoder.

It is not difficult to design small Bayesian networks with cycles for which probability propagation is unstable. Is there a way to easily distinguish between those graphs for which propagation will work and those graphs for which propagation is unstable? A belief that is not uncommon in the graphical models community is that short cycles are particularly apt to lead probability propagation astray. Although it is possible to design networks where this is so, there seems to be a variety of interesting networks

(such as the Hamming code network described above) for which propagation works well, despite short cycles.

The probability distributions that we deal with in decoding are very special distributions: the true posterior probability mass is actually concentrated in one microstate in a space of size $2^M$ where $M$ is large (*e.g.*, 10,000). The decoding problem is to find this most probable microstate, and it may be that iterative probability propagation decoders work because the true probability distribution is concentrated in this microstate.

We believe that there are many interesting and contentious issues in this area that remain to be resolved.

## Acknowledgements

We thank Frank Kschischang, Bob McEliece, and Radford Neal for discussions related to this work, and Zoubin Ghahramani for comments on a draft of this paper. This research was supported in part by grants from the Gatsby foundation, the Information Technology Research Council, and the Natural Sciences and Engineering Research Council.

## References

C. Berrou and A. Glavieux 1996. Near optimum error correcting coding and decoding: Turbo-codes. *IEEE Transactions on Communications* **44**, 1261–1271.

G. F. Cooper 1990. The computational complexity of probabilistic inference using Bayesian belief networks. *Artificial Intelligence* **42**, 393–405.

B. J. Frey 1998. *Graphical Models for Machine Learning and Digital Communication*, MIT Press, Cambridge, MA. See http://www.cs.utoronto.ca/~frey.

R. G. Gallager 1963. *Low-Density Parity-Check Codes*, MIT Press, Cambridge, MA.

S. Lin and D. J. Costello, Jr. 1983. *Error Control Coding: Fundamentals and Applications*, Prentice-Hall Inc., Englewood Cliffs, NJ.

S. L. Lauritzen and D. J. Spiegelhalter 1988. Local computations with probabilities on graphical structures and their application to expert systems. *Journal of the Royal Statistical Society B* **50**, 157–224.

D. J. C. MacKay and R. M. Neal 1995. Good codes based on very sparse matrices. In *Cryptography and Coding. 5th IMA Conference*, number 1025 in Lecture Notes in Computer Science, 100-111, Springer, Berlin Germany.

D. J. C. MacKay and R. M. Neal 1996. Near Shannon limit performance of low density parity check codes. *Electronics Letters* **32**, 1645–1646. Due to editing errors, reprinted in *Electronics Letters* **33**, 457–458.

J. Pearl 1988. *Probabilistic Reasoning in Intelligent Systems*, Morgan Kaufmann, San Mateo, CA.

C. E. Shannon 1948. A mathematical theory of communication. *Bell System Technical Journal* **27**, 379–423, 623–656.

P. Smyth, D. Heckerman, and M. I. Jordan 1997. Probabilistic independence networks for hidden Markov probability models. *Neural Computation* **9**, 227–270.